# Coarticulation in Markov Decision Processes

**Khashayar Rohanimanesh**
Department of Computer Science
University of Massachusetts
Amherst, MA 01003
*khash@cs.umass.edu*

**Robert Platt**
Department of Computer Science
University of Massachusetts
Amherst, MA 01003
*rplatt@cs.umass.edu*

**Sridhar Mahadevan**
Department of Computer Science
University of Massachusetts
Amherst, MA 01003
*mahadeva@cs.umass.edu*

**Roderic Grupen**
Department of Computer Science
University of Massachusetts
Amherst, MA 01003
*grupen@cs.umass.edu*

## Abstract

We investigate an approach for simultaneously committing to multiple activities, each modeled as a temporally extended action in a semi-Markov decision process (SMDP). For each activity we define a set of admissible solutions consisting of the redundant set of optimal policies, and those policies that ascend the optimal state-value function associated with them. A plan is then generated by merging them in such a way that the solutions to the subordinate activities are realized in the set of admissible solutions satisfying the superior activities. We present our theoretical results and empirically evaluate our approach in a simulated domain.

## 1 Introduction

Many real-world planning problems involve concurrent optimization of a set of prioritized subgoals of the problem by dynamically merging a set of (previously learned) policies optimizing the subgoals. A familiar example of this type of problem would be a driving task which may involve subgoals such as safely navigating the car, talking on the cell phone, and drinking coffee, with the first subgoal taking precedence over the others. In general this is a challenging problem, since activities often have conflicting objectives and compete for limited amount of resources in the system.

We refer to the behavior of an agent that simultaneously commits to multiple objectives as *Coarticulation*, inspired by the coarticulation phenomenon in speech. In this paper we investigate a framework based on semi-Markov decision processes (SMDPs) for studying this problem. We assume that the agent has access to a set of learned activities modeled by a set of SMDP *controllers* $\zeta = \{\mathcal{C}_1, \mathcal{C}_2, \ldots, \mathcal{C}_n\}$ each achieving a subgoal $\omega_i$ from a set of subgoals $\Omega = \{\omega_1, \omega_2, \ldots, \omega_n\}$. We further assume that the agent-environment interaction is an episodic task where at the be-

ginning of each episode a subset of subgoals $\omega \subseteq \Omega$ are introduced to the agent, where subgoals are ranked according to some priority ranking system. The agent is to devise a global policy by merging the policies associated with the controllers into a global policy that simultaneously commits to them according to their degree of significance. In general optimal policies of controllers do not offer flexibility required for the merging process. Thus for every controller we also compute a set of *admissible* suboptimal policies that reflect the degree of flexibility we can afford in it. Given a controller, an admissible policy is either an optimal policy, or it is a policy that *ascends* the optimal state-value function associated with the controller (i.e., in average leads to states with higher values), and is not too off from the optimal policy. To illustrate this idea, consider Figure 1(a) that shows a two dimensional

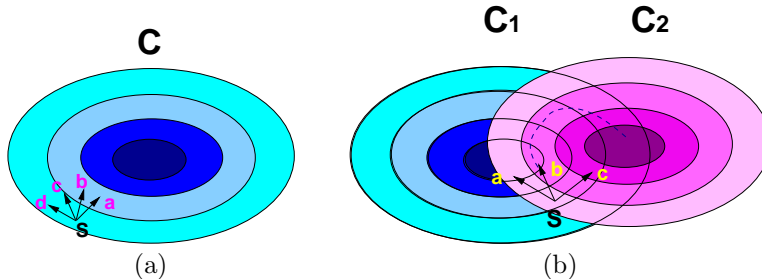

(a)  (b)

Figure 1: (a) actions $a$, $b$, and $c$ are ascending on the state-value function associated with the controller $\mathcal{C}$, while action $d$ is descending; (b) action $a$ and $c$ ascend the state-value function $\mathcal{C}_1$ and $\mathcal{C}_2$ respectively, while they descend on the state-value function of the other controller. However action $b$ ascends the state-value function of both controllers.

state-value function. Regions with darker colors represents states with higher values. Assume that the agent is currently in state marked $s$. The arrows show the direction of state transition as a result of executing different actions, namely actions $a$, $b$, $c$, and $d$. The first three actions lead the agent to states with higher values, in other words they ascend the state-value function, while action $d$ descends it. Figure 1(b) shows how introducing admissible policies enables simultaneously solving multiple subgoals. In this figure, action $a$ and $c$ are optimal in controllers $\mathcal{C}_1$ and $\mathcal{C}_2$ respectively, but they both descend the state-value function of the other controller. However if we allow actions such as action $b$, we are guaranteed to ascend both value functions, with a slight degeneracy in optimality.

Most of the related work in the context of MDPs assume that the subprocesses modeling the activities are additive utility independent [1, 2] and do not address concurrent planning with temporal activities. In contrast we focus on problems that involve temporal abstraction where the overall utility function may be expressed as a non-linear function of sub-utility functions that have different priorities. Our approach is also similar in spirit to the redundancy utilization formalism in robotics [4, 3, 6]. Most of these ideas, however, have been investigated in continuous domains and have not been extended to discrete domains. In contrast we focus on discrete domains modeled as MDPs.

In this paper we formally introduce the framework of redundant controllers in terms of the set of admissible policies associated with them and present an algorithm for merging such policies given a coarticulation task. We also present a set of theoretical results analyzing various properties of such controllers, and also the performance of the policy merging algorithm. The theoretical results are complemented by an experimental study that illustrates the trade-offs between the degree of flexibility of controllers and the performance of the policy generated by the merging process.

## 2  Redundant Controllers

In this section we introduce the framework of redundant controllers and formally define the set of admissible policies in them. For modeling controllers, we use the concept of subgoal options [7]. A subgoal option can be viewed as a closed loop controller that achieves a subgoal of some kind. Formally, a subgoal option of an MDP $M = \langle \mathcal{S}, \mathcal{A}, \mathcal{P}, \mathcal{R} \rangle$ is defined by a tuple $\mathcal{C} = \langle \mathcal{M}_\mathcal{C}, \mathcal{I}, \beta \rangle$. The MDP $\mathcal{M}_\mathcal{C} = \langle \mathcal{S}_\mathcal{C}, \mathcal{A}_\mathcal{C}, \mathcal{P}_\mathcal{C}, \mathcal{R}_\mathcal{C} \rangle$ is the option MDP induced by the option $\mathcal{C}$ in which $\mathcal{S}_\mathcal{C} \subseteq \mathcal{S}$, $\mathcal{A}_\mathcal{C} \subseteq \mathcal{A}$, $\mathcal{P}_\mathcal{C}$ is the transition probability function induced by $\mathcal{P}$, and $\mathcal{R}_\mathcal{C}$ is chosen to reflect the subgoal of the option. The policy component of such options are the solutions to the option MDP $\mathcal{M}_\mathcal{C}$ associated with them. For generality, throughout this paper we refer to subgoal options simply as controllers.

For theoretical reasons, in this paper we assume that each controller optimizes a minimum cost-to-goal problem. An MDP $\mathcal{M}$ modeling a minimum cost-to-goal problem includes a set of goal states $\mathcal{S}_\mathcal{G} \subset \mathcal{S}$. We also represent the set of non-goal states by $\bar{\mathcal{S}}_\mathcal{G} = \mathcal{S} - \mathcal{S}_\mathcal{G}$. Every action in a non-goal state incurs some negative reward and the agent receives a reward of zero in goal states. A controller $\mathcal{C}$ is a minimum cost-to-goal controller, if $\mathcal{M}_\mathcal{C}$ optimizes a minimum cost-to-goal problem. The controller also terminates with probability one in every goal state. We are now ready to formally introduce the concept of ascending policies in an MDP:

**Definition 1:** Given an MDP $\mathcal{M} = \langle \mathcal{S}, \mathcal{A}, \mathcal{P}, \mathcal{R} \rangle$, a function $\mathcal{L} : \mathcal{S} \rightarrow \mathbb{R}$, and a deterministic policy $\pi : \mathcal{S} \rightarrow \mathcal{A}$, let $\rho^\pi(s) = E_{s' \sim \mathcal{P}_s^{\pi(s)}} \{ \mathcal{L}(s') \} - \mathcal{L}(s)$, where $E_{s' \sim \mathcal{P}_s^{\pi(s)}} \{ . \}$ is the expectation with respect to the distribution over next states given the current state and the policy $\pi$. Then $\pi$ is *ascending* on $\mathcal{L}$, if for every state $s$ (except for the goal states if the MDP models a minimum cost-to-goal problem) we have $\rho^\pi(s) > 0$.

For an ascending policy $\pi$ on a function $\mathcal{L}$, function $\rho : \mathcal{S} \rightarrow \mathbb{R}^+$ gives a strictly positive value that measures how much the policy $\pi$ ascends on $\mathcal{L}$ in state $s$. A deterministic policy $\pi$ is descending on $\mathcal{L}$, if for some state $s$, $\rho^\pi(s) < 0$. In general we would like to study how a given policy behaves with respect to the optimal value function in a problem. Thus we choose the function $\mathcal{L}$ to be the optimal state value function (i.e., $\mathcal{V}^*$). The above condition can be interpreted as follows: we are interested in policies that in average lead to states with higher values, or in other words *ascend* the state-value function surface. Note that Definition 1 is closely related to the Lyapunov functions introduced in [5]. The minimum and maximum rate at which an ascending policy in average ascends $\mathcal{V}^*$ are given by:

**Definition 2:** Assume that the policy $\pi$ is ascending on the optimal state value function $\mathcal{V}^*$. Then $\pi$ ascends on $\mathcal{V}^*$ with a factor at least $\alpha$, if for all non-goal states $s \in \bar{\mathcal{S}}_\mathcal{G}$, $\rho^\pi(s) \geq \alpha > 0$. We also define the guaranteed expected ascend rate of $\pi$ as: $\kappa^\pi = \min_{s \in \bar{\mathcal{S}}_\mathcal{G}} \rho^\pi(s)$. The maximum possible achievable expected ascend rate of $\pi$ is also given by $\eta^\pi = \max_{s \in \bar{\mathcal{S}}_\mathcal{G}} \rho^\pi(s)$.

One problem with ascending policies is that Definition 1 ignores the immediate reward which the agent receives. For example it could be the case that as a result of executing an ascending policy, the agent transitions to some state with a higher value, but receives a huge negative reward. This can be counterbalanced by adding a second condition that keeps the ascending policies close to the optimal policy:

**Definition 3:** Given a minimum cost-to-goal problem modeled by an MDP $\mathcal{M} = \langle \mathcal{S}, \mathcal{A}, \mathcal{P}, \mathcal{R} \rangle$, a deterministic policy $\pi$ is $\epsilon$-ascending on $\mathcal{M}$ if: (1) $\pi$ is ascending on $\mathcal{V}^*$, and (2) $\epsilon$ is the maximum value in the interval $(0, 1]$ such that $\forall s \in \mathcal{S}$ we have $\mathcal{Q}^*(s, \pi(s)) \geq \frac{1}{\epsilon} \mathcal{V}^*(s)$.

Here, $\epsilon$ measures how close the ascending policy $\pi$ is to the optimal policy. For any $\epsilon$, the second condition assures that: $\forall s \in \mathcal{S}, \mathcal{Q}^*(s, \pi(s)) \in [\frac{1}{\epsilon} \mathcal{V}^*(s), \mathcal{V}^*(s)]$ (note

that because $\mathcal{M}$ models a minimum cost-to-goal problem, all values are negative). Naturally we often prefer policies that are $\epsilon$-ascending for $\epsilon$ values close to 1. In section 3 we derive a lower bound on $\epsilon$ such that no policy for values smaller than this bound is ascending on $\mathcal{V}^*$ (in other words $\epsilon$ cannot be arbitrarily small). Similarly, a deterministic policy $\pi$ is called $\epsilon$-ascending on $\mathcal{C}$, if $\pi$ is $\epsilon$-ascending on $\mathcal{M}_\mathcal{C}$. Next, we introduce the framework of redundant controllers:

**Definition 4:** A minimum cost-to-goal controller $\mathcal{C}$ is an $\epsilon$-*redundant* controller if there exist multiple deterministic policies that are either optimal, or $\epsilon$-ascending on $\mathcal{C}$. We represent the set of such admissible policies by $\chi_\mathcal{C}^\epsilon$. Also, the minimum ascend rate of $\mathcal{C}$ is defined as: $\bar{\kappa} = \min_{\pi \in \chi_\mathcal{C}^\epsilon} \kappa^\pi$, where $\kappa^\pi$ is the ascend rate of a policy $\pi \in \chi_\mathcal{C}^\epsilon$ (see Definition 2).

We can compute the $\epsilon$-redundant set of policies for a controller $\mathcal{C}$ as follows. Using the reward model, state transition model, $\mathcal{V}^*$ and $\mathcal{Q}^*$, in every state $s \in S$, we compute the set of actions that are $\epsilon$-ascending on $\mathcal{C}$ represented by $\mathcal{A}_\mathcal{C}^\epsilon(s) = \{a \in \mathcal{A} \mid a = \pi(s), \pi \in \chi_\mathcal{C}^\epsilon\}$, that satisfy both conditions of Definition 2.

Next, we present an algorithm for merging policies associated with a set of prioritized redundant controllers that run in parallel. For specifying the order of priority relation among the controllers we use the expression $\mathcal{C}_j \lhd \mathcal{C}_i$, where the relation "$\lhd$" expresses the *subject-to* relation (taken from [3]). This equation should read: controller $\mathcal{C}_j$ subject-to controller $\mathcal{C}_i$. A priority ranking system is then specified by a set of relations $\{\mathcal{C}_j \lhd \mathcal{C}_i\}$. Without loss of generality we assume that the controllers are prioritized based on the following ranking system: $\{\mathcal{C}_j \lhd \mathcal{C}_i \mid i < j\}$. Algorithm *MergeController* summarizes the policy merging process. In this algo-

---

**Algorithm 1** Function MergeController(s, $\mathcal{C}_1, \mathcal{C}_3, \ldots, \mathcal{C}_m$)

---

1: Input: current state $s$; the set of controllers $\mathcal{C}_i$; the redundant-sets $\mathcal{A}_{\mathcal{C}_i}^{\epsilon_i}(s)$ for every controller $\mathcal{C}_i$.
2: Initialize: $\Lambda_1(s) = \mathcal{A}_{\mathcal{C}_1}^{\epsilon_1}(s)$.
3: For $i = 2, 3, \ldots, n$ perform:
   $\Lambda_i(s) = \{a \mid a \in \mathcal{A}_{\mathcal{C}_i}^{\epsilon_i}(s) \wedge a \in \Lambda_{f(i)}(s)\}$ where $f(i) = \max j < i$ such that $\Lambda_j(s) \neq \emptyset$ (initially $f(1) = 1$).
4: Return an action $a \in \Lambda_{f(n+1)}(s)$.

---

rithm, $\Lambda_i(s)$ represents the ordered intersection of the redundant-sets $\mathcal{A}_{\mathcal{C}_j}^{\epsilon_j}$ up to the controller $\mathcal{C}_i$ (i.e., $1 \leq j \leq i$) constrained by the order of priority. In other words, each set $\Lambda_i(s)$ contains a set of actions in state $s$ that are all $\epsilon_i$-ascending with respect to the superior controllers $\mathcal{C}_1, \mathcal{C}_2, \ldots, \mathcal{C}_i$. Due to the limited amount of redundancy in the system, it is possible that the system may not be able to commit to some of the subordinate controllers. This happens when none of the actions with respect to some controller $\mathcal{C}_j$ (i.e., $a \in \mathcal{A}_{\mathcal{C}_j}^{\epsilon_j}(s)$) are $\epsilon$-ascending with respect to the superior controllers. In this case the algorithm skips the controller $\mathcal{C}_j$, and continues the search in the redundant-sets of the remaining subordinate controllers. The complexity of the above algorithm consists of the following costs: (1) cost of computing the redundant-sets $\mathcal{A}_{\mathcal{C}_i}^{\epsilon_i}$ for a controller which is linear in the number of states and actions: $O(|S| |A|)$, (2) cost of performing Algorithm *MergeController* in every state $s$, which is $O((m-1) |\mathcal{A}|^2)$, where $m$ is the number of subgoals. In the next section, we theoretically analyze redundant controllers and the performance of the policy merging algorithm in various situations.

# 3   Theoretical Results

In this section we present some of our theoretical results characterizing $\epsilon$-redundant controllers, in terms of the bounds on the number of time steps it takes for a controller to complete its task, and the performance of the policy merging algorithm. For lack of space, we have left out the proofs and refer the readers to [8]. In section 2 we stated that there is a lower bound on $\epsilon$ such that there exist no $\epsilon$-ascending policy for values smaller than this bound. In the first theorem we compute this lower bound:

**Theorem 1** Let $\mathcal{M} = \langle \mathcal{S}, \mathcal{A}, \mathcal{P}, \mathcal{R} \rangle$ be a minimum cost-to-goal MDP and let $\pi$ be an $\epsilon$-ascending policy defined on $\mathcal{M}$. Then $\epsilon$ is bounded by $\epsilon > \frac{|\mathcal{V}^*_{max}|}{|\mathcal{V}^*_{min}|}$, where $\mathcal{V}^*_{min} = \min_{s \in \bar{\mathcal{S}}_{\mathcal{G}}} \mathcal{V}^*(s)$ and $\mathcal{V}^*_{max} = \max_{s \in \bar{\mathcal{S}}_{\mathcal{G}}} \mathcal{V}^*(s)$.

Such a lower bound characterizes the maximum flexibility we can afford in a redundant controller and gives us an insight on the range of $\epsilon$ values that we can choose for it. In the second theorem we derive an upper bound on the expected number of steps that a minimum cost-to-goal controller takes to complete when executing an $\epsilon$-ascending policy:

**Theorem 2** Let $\mathcal{C}$ be an $\epsilon$-ascending minimum cost-to-goal controller and let $s$ denote the current state of the controller. Then any $\epsilon$-ascending policy $\pi$ on $\mathcal{C}$ will terminate the controller in some goal state with probability one. Furthermore, termination occurs in average in at most $\lceil \frac{-\mathcal{V}^*(s)}{\kappa^\pi} \rceil$ steps, where $\kappa^\pi$ is the guaranteed expected ascend rate of the policy $\pi$.

This result assures that the controller arrives in a goal state and will achieve its goal in a bounded number of steps. We use this result when studying performance of running multiple redundant controllers in parallel. Next, we study how concurrent execution of two controllers using Algorithm *MergeController* impacts each controller (this result can be trivially extended to the case when a set of $m > 2$ controllers are executed concurrently):

**Theorem 3** Given an MDP $\mathcal{M} = \langle \mathcal{S}, \mathcal{A}, \mathcal{P}, \mathcal{R} \rangle$, and any two minimum cost-to-goal redundant controllers $\{\mathcal{C}_1, \mathcal{C}_2\}$ defined over $\mathcal{M}$, the policy $\pi$ obtained by Algorithm *MergeController* based on the ranking system $\{\mathcal{C}_2 \lhd \mathcal{C}_1\}$ is $\epsilon_1$-ascending on $\mathcal{C}_1(s)$. Moreover, if $\forall s \in \mathcal{S}, \mathcal{A}^{\epsilon_1}_{\mathcal{C}_1}(s) \cap \mathcal{A}^{\epsilon_2}_{\mathcal{C}_2}(s) \neq \emptyset$, policy $\pi$ will be ascending on both controllers with the ascend rate at least $\kappa^\pi = \min\{\kappa^{\pi_1}, \kappa^{\pi_2}\}$.

This theorem states that merging policies of two controllers using Algorithm *MergeController* would generate a policy that remains $\epsilon_1$-ascending on the superior controller. In other words it does not negatively impact the superior controller. In the next theorem, we establish bounds on the expected number of steps that it takes for the policy obtained by Algorithm *MergeController* to achieve a set of prioritized subgoals $\omega = \{\omega_1, \ldots, \omega_m\}$ by concurrently executing the associated controllers $\{\mathcal{C}_1, \ldots, \mathcal{C}_m\}$:

**Theorem 4** Assume $\zeta = \{\mathcal{C}_1, \mathcal{C}_2, \ldots, \mathcal{C}_m\}$ is a set of minimum cost-to-goal $\epsilon_i$-redundant $(i = 1, \ldots, m)$ controllers defined over MDP $\mathcal{M}$. Let the policy $\pi$ denote the policy obtained by Algorithm *MergeController* based on the ranking system $\{\mathcal{C}_j \lhd \mathcal{C}_i | i < j\}$. Let $\mu_\zeta(s)$ denote the expected number of steps for the policy $\pi$ for achieving all the subgoals $\{\omega_1, \omega_2, \ldots, \omega_m\}$ associated with the set of controllers, assuming that the current state of the system is $s$. Then the following expression holds:

$$\max_i \lceil \frac{-\mathcal{V}^*_i(s)}{\eta^\pi_i} \rceil \leq \mu_\zeta(s) \leq \sum_{h \in \mathcal{H}} \mathcal{P}(h) \sum_{i=1}^{m} \lceil \frac{-\mathcal{V}^*_i(h(i))}{\bar{\kappa}_i} \rceil \tag{1}$$

where $\eta^\pi_i$ is the maximum possible achievable expected ascend rate for the controller $\mathcal{C}_i$ (see Definition 2), $\mathcal{H}$ is the set of sequences $h = \langle s, g_1, g_2, \ldots, g_m \rangle$ in which $g_i$ is a goal state in controller $\mathcal{C}_i$ (i.e., $g_i \in \mathcal{S}_{\mathcal{G}_i}$). The probability distribution

$\mathcal{P}(h) = \mathcal{P}^{\mathcal{C}_1}_{sg_1} \prod_{i=2}^{m} \mathcal{P}^{\mathcal{C}_i}_{g_{i-1}g_i}$ over sequences $h \in \mathcal{H}$ gives the probability of executing the set of controllers in sequence based on the order of priority starting in state $s$, and observing the goal state sequence $\langle g_1, \ldots, g_m \rangle$.

Based on Theorem 3, when Algorithm *MergeController* always finds a policy $\pi$ that optimizes all controllers (i.e., $\forall s \in \mathcal{S}, \cap_{i=1}^{m} \mathcal{A}^{\epsilon_i}_{\mathcal{C}_i}(s) \neq \emptyset$), policy $\pi$ will ascend on all controllers. Thus in average the total time for all controllers to terminate equals the time required for a controller that takes the most time to complete which has the lower bound of $\max_i \lceil \frac{-\mathcal{V}^*_i(s)}{\eta^{\pi}(s)} \rceil$. The worst case happens when the policy $\pi$ generated by Algorithm *MergeController* can not optimize more than one controller at a time. In this case $\pi$ always optimizes the controller with the highest priority until its termination, then optimizes the second highest priority controller and continues this process to the end in a sequential manner. The right hand side of the inequality given by Equation 1 gives an upper bound for the expected time required for all controllers to complete when they are executed sequentially. The above theorem implicitly states that when Algorithm *MergeController* generates a policy that in average commits to more than one subgoal it potentially takes less number of steps to achieve all the subgoals, compared to a policy that sequentially achieves them according to their degree of significance.

## 4    Experiments

In this section we present our experimental results analyzing redundant controllers and the policy merging algorithm described in section 2. Figure 2(a) shows a $10 \times 10$ grid world where an agent is to visit a set of prioritized locations marked by $G_1, \ldots, G_m$ (in this example $m = 4$). The agent's goal is to achieve all of the subgoals by focusing on superior subgoals and coarticulating with the subordinate ones. Intuitively, when the agent is navigating to some subgoal $G_i$ of higher priority, if some subgoal of lower priority $G_j$ is en route to $G_i$, or not too off from the optimal path to $G_i$, the agent may choose to visit $G_j$. We model this problem by an MDP

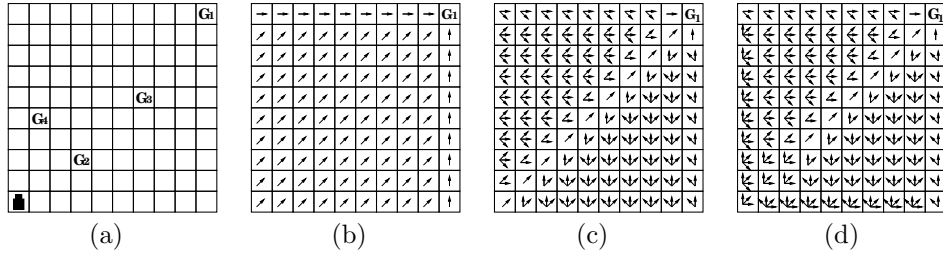

|     (a)     |     (b)     |     (c)     |     (d)     |

Figure 2: (a) A $10 \times 10$ grid world where an agent is to visit a set of prioritized subgoal locations; (b) The optimal policy associated with the subgoal $G_1$; (c) The $\epsilon$-ascending policy for $\epsilon = 0.95$; (d) The $\epsilon$-ascending policy for $\epsilon = 0.90$.

$\mathcal{M} = \langle \mathcal{S}, \mathcal{A}, \mathcal{R}, \mathcal{P} \rangle$, where $\mathcal{S}$ is the set of states consisting of 100 locations in the room, and $\mathcal{A}$ is the set of actions consisting of eight stochastic navigation actions (four actions in the compass direction, and four diagonal actions). Each action moves the agent in the corresponding direction with probability $p$ and fails with probability $(1 - p)$ (in all of the experiments we used success probability $p = 0.9$). Upon failure the agent is randomly placed in one of the eight-neighboring locations with equal probability. If a movement would take the agent into a wall, then the agent will remain in the same location. The agent also receives a reward of $-1$ for every action executed. We assume that the gent has access to a set of controllers $\mathcal{C}_1, \ldots, \mathcal{C}_m$, associated with the set of subgoal locations $G_1, \ldots, G_m$. A controller $\mathcal{C}_i$ is a minimum cost-to-goal subgoal option $\mathcal{C}_i = \langle \mathcal{M}_{\mathcal{C}_i}, \mathcal{I}, \beta \rangle$, where $\mathcal{M}_{\mathcal{C}_i} = \mathcal{M}$, the

initiation set $\mathcal{I}$ includes any locations except for the subgoal location, and $\beta$ forces the option to terminate only in the subgoal location. Figures 2(b)-(d) show examples of admissible policies for subgoal $G_1$: Figure 2(b) shows the optimal policy of the controller $\mathcal{C}_1$ (navigating the agent to the location $G_1$). Figures 2(c) and 2(d) show the $\epsilon$-redundant policies for $\epsilon = 0.95$ and $\epsilon = 0.90$ respectively. Note that by reducing $\epsilon$, we obtain a larger set of admissible policies although less optimal.

We use two different planning methods: (1) sequential planning, where we achieve the subgoals sequentially by executing the controllers one at a time according to the order of priority of subgoals, (2) concurrent planning, where we use Algorithm *MergeController* for merging the policies associated with the controllers. In the first set of experiments, we fix the number of subgoals. At the beginning of each episode the agent is placed in a random location, and a fixed number of subgoals (in our experiments $m = 4$) are randomly selected. Next, the set of admissible policies (using $\epsilon = 0.9$) for every subgoal is computed. Figure 3(a) shows the performance of both planning methods, for every starting location in terms of number of steps for completing the overall task. The concurrent planning method consistently outperforms the sequential planning in all starting locations. Next, for the

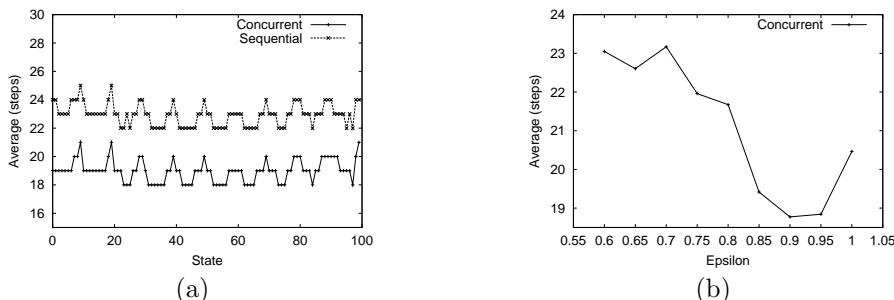

(a)    (b)

Figure 3: (a) Performance of both planning methods in terms of the average number of steps in every starting state; (b) Performance of the concurrent method for different values of $\epsilon$.

same task, we measure how the performance of the concurrent method varies by varying $\epsilon$, when computing the set of $\epsilon$-ascending policies for every subgoal. Figure 3(b) shows the performance of the concurrent method and Figure 4(a) shows the average number of subgoals coarticulated by the agent – averaged over all states – for different values of $\epsilon$. We varied $\epsilon$ from 0.6 to 1.0 using 0.05 intervals. All of these results are also averaged over 100 episodes, each consisting of 10 trials. Note that for $\epsilon = 1$, the only admissible policy is the optimal policy and thus it does not offer much flexibility with respect to the other subgoals. This can be seen in Figure 3(b) in which the policy generated by the merging algorithm for $\epsilon = 1.0$ has the minimum commitment to the other subgoals. As we reduce $\epsilon$, we obtain a larger set of admissible policies, thus we observe improvement in the performance. However, the more we reduce $\epsilon$, the less optimal admissible policies we obtain. Thus the performance degrades (here we can observe it for the values below $\epsilon = 0.85$). Figure 4(a) also shows by relaxing optimality (reducing $\epsilon$), the policy generated by the merging algorithm commits to more subgoals simultaneously.

In the final set of experiments, we fixed $\epsilon$ to 0.9 and varied the number of subgoals from $m = 2$ to $m = 50$ (all of these results are averaged over 100 episodes, each consisting of 10 trials). Figure 4(b) shows the performance of both planning methods. It can be observed that the concurrent method consistently outperforms the sequential method by increasing the number of subgoals (top curve shows the performance of the sequential method and bottom curve shows that of concurrent method). This is because when there are many subgoals, the concurrent planning

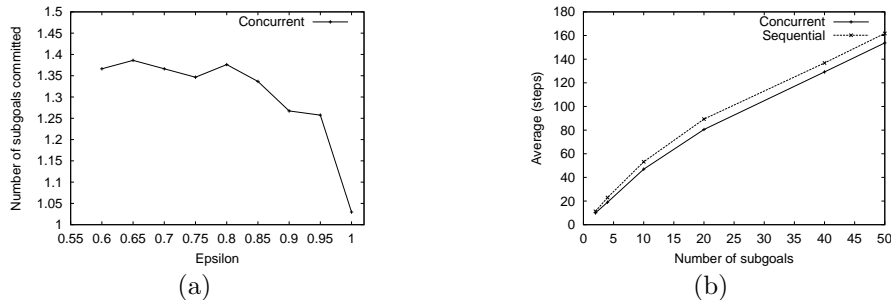

Figure 4: (a) Average number of subgoals coarticulated using the concurrent planning method for different values of $\epsilon$; (b) Performance of the planning methods in terms of the average number of steps in every starting state.

method is able to visit multiple subgoals of lower priority en route the primary subgoals, thus it can save more time.

## 5 Concluding Remarks

There are a number of questions and open issues that remain to be addressed and many interesting directions in which this work can be extended. In many problems, the strict order of priority of subtasks may be violated: in some situations we may want to be sub-optimal with respect to the superior subtasks in order to improve the overall performance. One other interesting direction is to study situations when actions are structured. We are currently investigating compact representation of the set of admissible policies by exploiting the structure of actions.

### Acknowledgements

This research is supported in part by a grant from the National Science Foundation #ECS-0218125.

## References

[1] C. Boutilier, R. Brafman, and C. Geib. Prioritized goal decomposition of Markov decision processes: Towards a synthesis of classical and decision theoretic planning. In Martha Pollack, editor, *Proceedings of the Fifteenth International Joint Conference on Artificial Intelligence*, pages 1156–1163, San Francisco, 1997. Morgan Kaufmann.

[2] C. Guestrin and G. Gordon. Distributed planning in hierarchical factored mdps. In *In the Proceedings of the Eighteenth Conference on Uncertainty in Artificial Intelligence*, pages 197 – 206, Edmonton, Canada, 2002.

[3] M. Huber. *A Hybrid Architecture for Adaptive Robot Control*. PhD thesis, University of Massachusetts, Amherst, 2000.

[4] Y. Nakamura. *Advanced robotics: redundancy and optimization*. Addison-Wesley Pub. Co., 1991.

[5] Theodore J. Perkins and Andrew G. Barto. Lyapunov-constrained action sets for reinforcement learning. In *Proc. 18th International Conf. on Machine Learning*, pages 409–416. Morgan Kaufmann, San Francisco, CA, 2001.

[6] R. Platt, A. Fagg, and R. Grupen. Nullspace composition of control laws for grasping. *In the Proceedings of the IEEE/RSJ International Conference on Intelligent Robots and Systems (IROS)*, 2002.

[7] D. Precup. *Temporal Abstraction in Reinforcement Learning*. PhD thesis, Department of Computer Science, University of Massachusetts, Amherst., 2000.

[8] K. Rohanimanesh, R. Platt, S. Mahadevan, and R. Grupen. A framework for coarticulation in markov decision processes. Technical Report 04-33, (`www.cs.umass.edu/~khash/coarticulation04.pdf`), Department of Computer Science, University of Massachusetts, Amherst, Massachusetts, USA., 2004.